# Learning from Data of Variable Quality

**Koby Crammer, Michael Kearns, Jennifer Wortman**
Computer and Information Science
University of Pennsylvania
Philadelphia, PA 19103
{crammer,mkearns,wortmanj}@cis.upenn.edu

## Abstract

We initiate the study of learning from multiple sources of limited data, each of which may be corrupted at a different rate. We develop a complete theory of which data sources should be used for two fundamental problems: estimating the bias of a coin, and learning a classifier in the presence of label noise. In both cases, efficient algorithms are provided for computing the optimal subset of data.

## 1 Introduction

In many natural machine learning settings, one is not only faced with data that may be corrupted or deficient in some way (classification noise or other label errors, missing attributes, and so on), but with data that is not *uniformly* corrupted. In other words, we might be presented with data of *variable quality* — perhaps some small amount of entirely "clean" data, another amount of slightly corrupted data, yet more that is significantly corrupted, and so on. Furthermore, in such circumstances we may often know at least an upper bound on the rate and type of corruption in each pile of data. An extreme example is the recent interest in settings where one has a very limited set of correctly labeled examples, and an effectively unlimited set of entirely unlabeled examples, as naturally arises in problems such as classifying web pages [1]. Another general category of problems that falls within our interest is when multiple piles of data are drawn from processes that differ perhaps slightly and in varying amounts from the process we wish to estimate. For example, we might wish to estimate a conditional distribution $P(X|Y=y)$ but have only a small number of observations in which $Y=y$, but a larger number of observations in which $Y=y'$ for values of $y'$ "near" to $y$. In such circumstances it might make sense to base our model on a larger number of observations, at least for those $y'$ closest to $y$.

While there is a large body of learning theory both for uncorrupted data and for data that is uniformly corrupted in some way [2, 3], there is no general framework and theory for learning from data of variable quality. In this paper we introduce such a framework, and develop its theory, for two basic problems: estimating a bias from corrupted coins, and learning a classifier in the presence of varying amounts of label noise. For the corrupted coins case we provide an upper bound on the error that is expressed as a trade-off between weighted approximation errors and larger amounts of data. This bound provides a building block for the classification noise setting, in which we are able to give a bound on the generalization error of empirical risk minimization that specifies the optimal subset of the

data to use. Both bounds can computed by simple and efficient algorithms. We illustrate both problems and our algorithms with numerical simulations.

## 2    Estimating the Bias from Corrupted Coins

We begin by considering perhaps the simplest possible instance of the general class of problems in which we are interested — namely, the problem of estimating the unknown bias of a coin. In this version of the variable quality model, we will have access to different amounts of data from "corrupted" coins whose bias differs from the one we wish to estimate. We use our solution for this simple problem as a building block for the classification noise setting in Section 3.

### 2.1    Problem Description

Suppose we wish to estimate the bias $\beta$ of a coin given $K$ piles of training observations $N_1, ..., N_K$. Each pile $N_i$ contains $n_i$ outcomes of flips of a coin with bias $\beta_i$, where the only information we are provided is that $\beta_i \in [\beta - \epsilon_i, \beta + \epsilon_i]$, and $0 \le \epsilon_1 \le \epsilon_2 \le ... \le \epsilon_K$. We refer to the $\epsilon_i$ as bounds on the *approximation errors* of the corrupted coins. We denote by $h_i$ the number of heads observed in the $i$th pile. Our immediate goal is to determine which piles should be considered in order to obtain the best estimate of the true bias $\beta$.

We consider estimates for $\beta$ obtained by merging some subset of the data into a single unified pile, and computing the maximum likelihood estimate for $\beta$, which is simply the fraction of times heads appears as an outcome in the unified pile. Although one can consider using *any* subset of the data, it can be proved (and is intuitively obvious) that an optimal estimate (in the sense that will be defined shortly) always uses a *prefix* of the data, i.e. all data from the piles indexed 1 to $k$ for some $k \le K$, and possibly a subset of the data from pile $k + 1$. In fact, it will be shown that only complete piles need to be considered. Therefore, from this point on we restrict ourselves to estimates of this form, and identify them by the maximal index $k$ of the piles used. The associated estimate is then simply

$$\hat{\beta}_k = \frac{h_1 + \ldots + h_k}{n_1 + \ldots + n_k} \ .$$

We denote the expectation of this estimate by

$$\bar{\beta}_k = \mathrm{E}\left[\hat{\beta}_k\right] = \frac{n_1\beta_1 + \ldots + n_k\beta_k}{n_1 + \ldots + n_k} \ .$$

To simplify the presentation we denote by $n_{i,j}$ the number of outcomes in piles $N_i, \ldots, N_j$, that is, $n_{i,j} = \sum_{m=i}^{j} n_m$ .

We now bound the deviation of the estimate $\hat{\beta}_k$ from the true bias of the coin $\beta$ using the expectation $\bar{\beta}_k$:

$$
\begin{aligned}
|\beta - \hat{\beta}_k| &= |\beta - \bar{\beta}_k + \bar{\beta}_k - \hat{\beta}_k| \\
&\le |\beta - \bar{\beta}_k| + |\bar{\beta}_k - \hat{\beta}_k| \\
&\le \sum_{i=1}^{k} \frac{n_i}{n_{1,k}}\epsilon_i + |\bar{\beta}_k - \hat{\beta}_k|
\end{aligned}
$$

The first inequality follows from the triangle inequality and the second from our assumptions. Using the Hoeffding inequality we can bound the second term and find that with high probability for an appropriate choice of $\delta$ we have

$$|\beta - \hat{\beta}_k| \le \sum_{i=1}^{k} \frac{n_i}{n_{1,k}}\epsilon_i + \sqrt{\frac{\log(2K/\delta)}{2n_{1,k}}} \ . \tag{1}$$

To summarize, we have proved the following theorem.

**Theorem 1** *Let $\hat{\beta}_k$ be the estimate obtained by using only the data from the first $k$ piles. Then for any $\delta > 0$, with probability $\geq 1 - \delta$ we have*

$$\left|\beta - \hat{\beta}_k\right| \leq \sum_{i=1}^{k} \frac{n_i}{n_{1,k}} \epsilon_i + \sqrt{\frac{\log(2K/\delta)}{2n_{1,k}}}$$

*simultaneously for all $k = 1, \ldots, K$.*

Two remarks are in place here. First, the theorem is data-independent since it does not take into account the actual outcomes of the experiments $h_1, \ldots, h_K$. Second, the two terms in the bound reflect the well-known trade-off between bias (approximation error) and variance (estimation error). The first term bounds the approximation error of replacing the true coin $\beta$ with the average $\bar{\beta}_k$. The second term corresponds to the estimation error which arises as a result of our finite sample size.

This theorem implies a natural algorithm to choose the number of piles $k^*$ as is the minimizer of the bound over the number of piles used:

$$k^* = \underset{k \in \{1,\ldots,K\}}{\text{argmin}} \left\{ \sum_{i=1}^{k} \frac{n_i}{n_{1,k}} \epsilon_i + \sqrt{\frac{\log(2K/\delta)}{2n_{1,k}}} \right\}.$$

To conclude this section we argue that our choice of using a prefix of piles is optimal. First, note that by adding a new pile with a corruption level $\epsilon$ smaller then the current corruption level, we can always reduce the bounds. Thus it is optimal to use prefix of the piles and not to ignore piles with low corruption levels. Second, we need to show that if we decide to use a pile, it will be optimal to use all of it. Note that we can choose to view each coin toss as a separate pile with a single observation, thus yielding $n_{1,K}$ piles of size 1. The following technical lemma states that under this view of singleton piles, once we decide to add a pile with some corruption level, it will be optimal to use all singleton piles with the same corruption level. The proof of this lemma is omitted due to lack of space.

**Lemma 1** *Assume that all the piles are of size $n_i = 1$ and that $\epsilon_k \leq \epsilon_{p+k} = \epsilon_{p+k+1}$. Then the following two inequalities cannot hold simultaneously:*

$$\sum_{i=1}^{k} \frac{n_i}{n_{1,k}} \epsilon_i + \sqrt{\frac{\log(2n_{1,K}/\delta)}{2n_{1,k}}} > \sum_{i=1}^{k+p} \frac{n_i}{n_{1,k+p}} \epsilon_i + \sqrt{\frac{\log(2n_{1,K}/\delta)}{2n_{1,k+p}}}$$

$$\sum_{i=1}^{k+p+1} \frac{n_i}{n_{1,k+p+1}} \epsilon_i + \sqrt{\frac{\log(2n_{1,K}/\delta)}{2n_{1,k+p+1}}} \geq \sum_{i=1}^{k+p} \frac{n_i}{n_{1,k+p}} \epsilon_i + \sqrt{\frac{\log(2n_{1,K}/\delta)}{2n_{1,k+p}}}.$$

In other words, if the bound on $|\beta - \hat{\beta}_{k+p}|$ is smaller than the bound on $|\beta - \hat{\beta}_k|$, then the bound on $|\beta - \hat{\beta}_{k+p+1}|$ must be smaller than both unless $\epsilon_{k+p+1} > \epsilon_{k+p}$. Thus if the $p$th and $p+1$th samples are from the same original pile (and $\epsilon_{k+p+1} = \epsilon_{k+p}$), then once we decide to use samples through $p$, we will always want to include sample $p + 1$. It follows that we must only consider using complete piles of data.

## 2.2 Corrupted Coins Simulations

The theory developed so far can be nicely illustrated via some simple simulations. We briefly describe just one such experiment in which there were $K = 8$ piles. The target coin was fair: $\beta = 0.5$. The approximation errors of the corrupted coins were

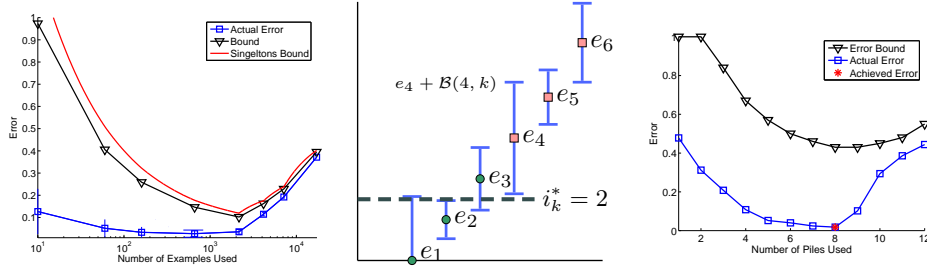

Figure 1: **Left:** Illustration of the actual error and our error bounds for estimating the bias of a coin. The error bars show one standard deviation. **Center:** Illustration of the interval construction. **Right:** Illustration of actual error of a 20 dimensional classification problem and the error bounds found using our methods.

$\vec{\epsilon} = (0.001, 0.01, 0.02, 0.03, 0.04, 0.2, 0.3, 0.5)$, and number of outcomes in the corresponding piles were $\vec{n} = (10, 50, 100, 500, 1500, 2000, 3000, 10000)$. The following process was repeated $1,000$ times. We set the probability of the $i$th coin to be $\beta_i = \beta + \epsilon_i$ and sampled $n_i$ times from it. We then used all possible prefixes $1, \ldots, k$ of piles to estimate $\beta$. For each $k$, we computed the bound for the estimate using piles $1, \ldots, k$ using the theory developed in the previous section. To illustrate Lemma 1 we also computed the bound using partial piles. This bound is slightly higher than the suggested bound since we use effectively more piles ($n_{1,K}$ instead of $K$). As the lemma predicts, it is not valuable to use subsets of piles. Simulations with other values of $K$, $\vec{\epsilon}$ and $\vec{n}$ yield similar qualitative behavior. We note that a strength of the theory developed is its generality, as it provides bounds for any model parameters.

The leftmost panel of Figure 1 summarizes the simulation results. Empirically, the best estimate of the target coin is using the first four piles, while our algorithm suggests using the first five piles. However, the empirical difference in quality between the two estimates is negligible, so the theory has given near-optimal guidance in this case. We note that while our bounds have essentially the right shape (which is what matters for the computation of $k^*$), numerically they are quite loose compared to the true behavior. There are various limits to the numerical precision we should expect without increasing the complexity of the theory — for example, the precision is limited by accuracy of constants in the Hoeffding inequality and the use of the union bound.

## 3   Classification with Label Noise

We next explore the problem of classification in the presence of multiple data sets with varying amounts of *label noise*. The setting is as follows. We assume there is a fixed and unknown binary function $f : X \to \{0, 1\}$ and a fixed and unknown distribution $P$ on the inputs $X$ to $f$. We are presented again with $K$ piles of data, $N_1, ..., N_K$. Now each pile $N_i$ contains $n_i$ labeled examples $(x, y)$ that are generated from the target function $f$ with label noise at rate $\eta_i$, where $0 \le \eta_1 < \eta_2 < ... < \eta_K$. In other words, for each example $(x, y)$ in pile $N_i$, $y = f(x)$ with probability $1 - \eta_i$ and $y = \neg f(x)$ with probability $\eta_i$. The goal is to decide which piles of data to use in order to choose a function $h$ from a set of hypothesis functions $\mathcal{H}$ with minimal *generalization (true) error* $e(h)$ with respect to $f$ and $P$. As before, for any prefix of piles $N_1, \ldots, N_k$, we examine the most basic estimator based on this data, namely the hypothesis minimizing the observed or training error:

$$\hat{h}_k = \underset{h \in \mathcal{H}}{\operatorname{argmin}} \{\hat{e}_k(h)\}$$

where $\hat{e}_k(h)$ is the fraction of times $h(x) \neq y$ over all $(x, y) \in N_1 \cup \cdots \cup N_k$. Thus we examine the standard empirical risk minimization framework [2]. Generalizing from the biased coin setting, we are interested in three primary questions: what can we say about the deviation $|e(\hat{h}_k) - \hat{e}(\hat{h}_k)|$, which is the gap between the true and observed error of the estimator $\hat{h}_k$; what is the optimal value of $k$; and how can we compute the corresponding bounds?

We note that the classification noise setting can naturally be viewed as a special case of a more general and challenging "agnostic" classification setting that we discuss briefly in Section 4. Here we provide a more specialized solution that exploits particular properties of class label noise.

We begin by observing that for any *fixed* function $h$, the question of how $\hat{e}_k(h)$ is related to $e(h)$ bears great similarity to the biased coin setting. More precisely, the expected classification error of $h$ on pile $N_i$ only is

$$(1 - \eta_i)e(h) + \eta_i(1 - e(h)) = e(h) + \eta_i(1 - 2e(h)) .$$

Thus if we set

$$\beta = e(h), \quad \epsilon_i = \eta_i \, |1 - 2e(h)| \tag{2}$$

and if we were only concerned with making the best use of the data in estimating $e(h)$, we could attempt to apply the theory developed in Section 2 using the reduction above. There are two distinct and obvious difficulties. The first difficulty is that even restricting attention to estimating $e(h)$ for a fixed $h$, the values for $\epsilon_i$ above (and thus the bounds computed by the methods of Section 2) depend on $e(h)$, which is exactly the unknown quantity we would like to estimate. The second difficulty is that in order to bound the performance of empirical error minimization within $\mathcal{H}$, we must say something about the probability of *any* $h \in \mathcal{H}$ being selected. We address each of these difficulties in turn.

### 3.1 Computing the Error Bound Matrix

For now we assume that $\{e(h) : h \in \mathcal{H}\}$ is a finite set containing $M$ values $e_1 < \ldots < e_M$. This assumption clearly holds if $|\mathcal{H}|$ is finite, and can be removed entirely by discretizing the values in $\{e(h) : h \in \mathcal{H}\}$. For convenience we assume that for all levels $e_i$ there exists a function $h \in \mathcal{H}$ such that $e(h) = e_i$. This assumption can also be removed (details of both omitted due to space considerations). We define a matrix $\mathcal{B}$ of estimation errors as follows. Each row $i$ of $\mathcal{B}$ represents one possible value of $e(h) = e_i$, while each column $k$ represents the use of only piles $N_1, \ldots, N_k$ of noisy labeled examples of the target $f$. The entry $\mathcal{B}(i, k)$ will contain a bound on $|e(h) - \hat{e}_k(h)|$ that is valid simultaneously for all $h \in \mathcal{H}$ with $e(h) = e_i$. In other words, for any such $h$, with high probability $\hat{e}_k(h)$ falls in the range $[e_i - \mathcal{B}(i, k), e_i + \mathcal{B}(i, k)]$. It is crucial to note that we do not need to know *which* functions $h \in \mathcal{H}$ satisfy $e(h) = e_i$ in order to either compute or use the bound $\mathcal{B}(i, k)$, as we shall see shortly. Rather, it is enough to know that for each $h \in \mathcal{H}$, some row of $\mathcal{B}$ will provide estimation error bounds for each $k$.

The values in $\mathcal{B}$ can be now be calculated using the settings provided by Eq. (2) and the bound in Eq. (1). However, since Eq. (1) applies to the case of a single biased coin and here we have many (essentially one for each function at a given generalization error $e_i$), we must modify it slightly. We can (pessimistically) bound the VC dimension of all functions with error rate $e(h) = e_i$ by the VC dimension $d$ of the entire class $\mathcal{H}$. Formally, we replace the square root term in Eq. (1) with the following expression, which is a simple application of VC theory [2, 3]:

$$\mathcal{O}\left( \sqrt{ \frac{1}{n_{1,k}} \left( d \log\left( \frac{n_{1,k}}{d} \right) + \log\left( \frac{KM}{\delta} \right) \right) } \right) . \tag{3}$$

We note that in cases where we have more information on the structure of the generalization errors in $\mathcal{H}$, an accordingly modified equation can be used, which may yield considerably improved bounds. For example, in the statistical physics theory of learning curves[4] it is common to posit knowledge of the density or number of functions in $\mathcal{H}$ at a given generalization error $e_i$. In such a case we could clearly substitute the VC dimension $d$ by the (potentially much smaller) VC dimension $d_i$ of just this subclass.

In a moment we describe how the matrix $\mathcal{B}$ can be used to choose the number $k$ of piles to use, and to compute a bound on the generalization error of $\hat{h}_k$. We first formalize the development above as an intermediate result.

**Lemma 2** *Suppose $\mathcal{H}$ is a set of binary functions with VC dimension $d$. Let $M$ be the number of noise levels and $K$ be the number of piles. Then for all $\delta > 0$, with probability at least $1 - \delta$, for all $i \in \{1, \ldots, M\}$, for all $h \in \mathcal{H}$ with $e(h) = e_i$, and for all $k \in \{1, \ldots, K\}$ we have*

$$|e(h) - \hat{e}_k(h)| \leq \mathcal{B}(i, k) .$$

*The matrix $\mathcal{B}$ can be computed in time linear in its size $\mathcal{O}(KM)$.*

### 3.2 Putting It All Together

By Lemma 2, the matrix $\mathcal{B}$ gives, for each possible generalization error $e_i$ and each $k$, an upper bound on the deviation between observed and true errors for functions of true error $e_i$ when using piles $N_1, \ldots, N_k$. It is thus natural to try to use column $k$ of $\mathcal{B}$ to bound the error of $\hat{h}_k$, the function minimizing the observed error on these piles.

Suppose we fix the number of piles used to be $k$. The observed error of any function with true generalization error $e_i$ must, with high probability, lie in the interval $I_{i,k} = [e_i - \mathcal{B}(i, k), e_i + \mathcal{B}(i, k)]$. By simultaneously considering these intervals for all values of $e_i$, we can put a bound on the generalization error of the best function in the hypothesis class. This process is best illustrated by an example.

Consider a hypothesis space in which the generalization error of the available functions can take on the discrete values 0, 0.1, 0.2, 0.3, 0.4, and 0.5. Suppose the matrix $\mathcal{B}$ has been calculated as above and the $k$th column is (0.16, 0.05, 0.08, 0.14, 0.07, 0.1). We know, for example, that all functions with true generalization error $e_2 = 0.1$ will show an error in the range $I_{2,k} = [0.05, 0.15]$, and that all functions with true generalization error $e_4 = 0.3$ will show an error in the range $I_{4,k} = [0.16, 0.44]$. The center panel of Figure 1 illustrates the span of each interval.

Examining this diagram, it becomes clear that the function $\hat{h}_k$ minimizing the error on $N_1 \cup \cdots \cup N_k$ could not possibly be a function with true error $e_4$ or higher as long as $\mathcal{H}$ contains at least one function with true error $e_2$ since the observed error of the latter would necessarily be lower (with high probability). Likewise, it would not be possible for a function with true error $e_5$ or $e_6$ to be chosen. However, a function with true error $e_3$ could produce a lower observed error than one with true error $e_1$ or $e_2$ (since $e_3 - \mathcal{B}(3, k) < e_2 + \mathcal{B}(2, k)$ and $e_3 - \mathcal{B}(3, k) < e_1 + \mathcal{B}(1, k)$), and thus could be chosen as $\hat{h}_k$. Therefore, the smallest bound we can place on the true error of $\hat{h}_k$ in this example is $e_3 = 0.2$.

In general, we know that $\hat{h}_k$ will have true error corresponding to the midpoint of an a interval which overlaps with the interval with the least upper bound ($I_{2,k}$ in this example). This leads to an intuitive procedure for calculating a bound on the true error of $\hat{h}_k$. First, we determine the interval with the smallest upper bound, $i_k^* = \text{argmin}_i\{e_i + \mathcal{B}(i, k)\}$. Consider the set of intervals which overlap with $i_k^*$, namely $J_k = \{i : e_i - \mathcal{B}(i, k) \leq e_{i_k^*} + \mathcal{B}(i_k^*, k)\}$. It is possible for the smallest observed error to come from a function corresponding to any

of the intervals in $J_k$. Thus, a bound on the true error of $\hat{h}_k$ can be obtained by taking the maximum $e(h)$ value for any function in $J_k$, i.e. $\mathcal{C}(k) \stackrel{\text{def}}{=} \max_{i \in J_k}\{e_i\}$.

Our overall algorithm for bounding $e(\hat{h}_k)$ and choosing $k^*$ can thus be summarized:

1. Compute the matrix $\mathcal{B}$ as described in Section 3.1 .
2. Compute the vector $\mathcal{C}$ described above.
3. Output $k^* = \operatorname{argmin}_k\{\mathcal{C}(k)\}$.

We have established the following theorem.

**Theorem 2** *Suppose $\mathcal{H}$ is a set of binary functions with VC dimension $d$. Let $M$ be the number of noise levels and $K$ be the number of piles. For all $k = 1, ..., K$, let $\hat{h}_k = \operatorname{argmin}_h\{\hat{e}_k(h)\}$ be the function in $\mathcal{H}$ with the lowest empirical error evaluated using the first $k$ piles of data. Then for all $\delta > 0$, with probability at least $1 - \delta$,*

$$e(\hat{h}_k) \leq \mathcal{C}(k)$$

*The suggested choice of $k$ is thus $k^* = \operatorname{argmin}_k\{\mathcal{C}(k)\}$.*

### 3.3 Classification Noise Simulations

In order to illustrate the methodology described in this section, simulations were run on a classification problem in which samples $\vec{x} \in \{0, 1\}^{20}$ were chosen uniformly at random, and the target function $f(\vec{x})$ was 1 if and only if $\sum_{i=1}^{20} x_i > 10$.

Classification models were created for $k = 1, ..., K$ by training using the first $k$ piles of data using logistic regression with a learning rate of $0.0005$ for a maximum of $5,000$ iterations. The generalization error for each model was determined by testing on a noise-free sample of 500 examples drawn from the same uniform distribution. Bounds were calculated using the algorithm described above with functions binned into 101 evenly spaced error values $\vec{e} = (0, 0.01, 0.02, ..., 1)$ with $\delta = 0.001$.

The right panel of Figure 1 shows an example of the bounds found with $K = 12$ piles, noise levels $\vec{\eta} = (0.001, 0.002, 0.01, 0.02, 0.03, 0.04, 0.05, 0.1, 0.2, 0.3, 0.4, 0.5)$, and sample sizes $\vec{n} = (20, 150, 300, 400, 500, 600, 700, 1000, 1500, 2000, 3000, 5000)$. The algorithm described above correctly predicts that the eighth pile should be chosen as the cutoff, yielding an optimal error value of $0.018$. It is interesting to note that although the error bounds shown are significantly higher than the actual error, the shapes of the curves are similar. This phenomena is common to many uniform convergence bounds.

Further experimentation has shown that the algorithm described here works well in general when there are small piles of low noise data and large piles of high noise data. Its predictions are more useful in higher dimensional space, since it is relatively easy to get good predictions without much available data in lower dimensions.

## 4  Further Research

In research subsequent to the results presented here [5], we examine a considerably more general "agnostic" classification setting [6]. As before, we assume there is a fixed and unknown binary function $f : X \rightarrow \{0, 1\}$ and a fixed and unknown distribution $P$ on the inputs $X$ to $f$. We are presented again with $K$ piles of data, $N_1, ..., N_K$. Now each pile $N_i$ contains $n_i$ labeled examples $(x, y)$ that are generated from an unknown function $h_i$ such that $e(h_i) = e(h_i, f) = \operatorname{Pr}_P[h_i(x) \neq f(x)] \leq \epsilon_i$ for given values $\epsilon_1 \leq \ldots \leq \epsilon_K$. Thus

we are provided piles of labeled examples of unknown functions "nearby" the unknown target $f$, where "nearby" is quantified by the sequence of $\epsilon_i$.

In forthcoming work [5] we show that with high probability, for any $k \leq K$

$$e(\hat{h}_k, f) \leq \min_{h \in \mathcal{H}} \{e(f, h)\} + 2 \sum_{i=1}^{k} \left( \frac{n_i}{n_{1,k}} \right) \epsilon_i + \mathcal{O} \left( \sqrt{\frac{1}{n_{1,k}} \left( d \log \left( \frac{n_{1,k}}{d} \right) + \log \left( \frac{K}{\delta} \right) \right)} \right)$$

This result again allows us to express the optimal number of piles as a trade-off between weighted approximation errors and increasing sample size. We suspect the result can be extended to a wider class of loss functions that just classification.

## References

[1] A. Blum and T. Mitchell. Combining labeled and unlabeled data with co-training. In *Proceedings of the Eleventh Annual Conference on Computational Learning Theory*, pages 92–100, 1998.

[2] V. N. Vapnik. *Statistical Learning Theory*. Wiley, 1998.

[3] M. J. Kearns and U. V. Vazirani. *An Introduction to Computational Learning Theory*. MIT Press, 1994.

[4] D. Haussler, M. Kearns, H.S. Seung, and N. Tishby. Rigorous learning curve bounds from statistical mechanics. In *Proceedings of the Seventh Annual ACM Conference on Computational Learning Theory*, pages 76–87, 1994.

[5] K. Crammer, M. Kearns, and J. Wortman. Forthcoming. 2006.

[6] M. Kearns, R. Schapire, and L. Sellie. Towards efficient agnostic learning. *Machine Learning*, 17:115–141, 1994.
